# Parallel Inference for Latent Dirichlet Allocation on Graphics Processing Units

**Feng Yan**
Department of CS
Purdue University
West Lafayette, IN 47907

**Ningyi Xu**
Microsoft Research Asia
No. 49 Zhichun Road
Beijing, P.R. China

**Yuan (Alan) Qi**
Departments of CS and Statistics
Purdue University
West Lafayette, IN 47907

## Abstract

The recent emergence of Graphics Processing Units (GPUs) as general-purpose parallel computing devices provides us with new opportunities to develop scalable learning methods for massive data. In this work, we consider the problem of parallelizing two inference methods on GPUs for latent Dirichlet Allocation (LDA) models, collapsed Gibbs sampling (CGS) and collapsed variational Bayesian (CVB). To address limited memory constraints on GPUs, we propose a novel data partitioning scheme that effectively reduces the memory cost. This partitioning scheme also balances the computational cost on each multiprocessor and enables us to easily avoid memory access conflicts. We use data streaming to handle extremely large datasets. Extensive experiments showed that our parallel inference methods consistently produced LDA models with the same predictive power as sequential training methods did but with 26x speedup for CGS and 196x speedup for CVB on a GPU with 30 multiprocessors. The proposed partitioning scheme and data streaming make our approach scalable with more multiprocessors. Furthermore, they can be used as general techniques to parallelize other machine learning models.

## 1   Introduction

Learning from massive datasets, such as text, images, and high throughput biological data, has applications in various scientific and engineering disciplines. The scale of these datasets, however, often demands high, sometimes prohibitive, computational cost. To address this issue, an obvious approach is to parallelize learning methods with multiple processors. While large CPU clusters are commonly used for parallel computing, Graphics Processing Units (GPUs) provide us with a powerful alternative platform for developing parallel machine learning methods.

A GPU has massively built-in parallel thread processors and high-speed memory, therefore providing potentially one or two magnitudes of peak flops and memory throughput greater than its CPU counterpart. Although GPU is not good at complex logical computation, it can significantly reduce running time of numerical computation-centric applications. Also, GPUs are more *cost effective* and *energy efficient*. The current high-end GPU has over 50x more peak flops than CPUs at the same price. Given a similar power consumption, GPUs perform more flops per watt than CPUs. For large-scale industrial applications, such as web search engines, efficient learning methods on GPUs can make a big difference in energy consumption and equipment cost. However, parallel computing

on GPUs can be a challenging task because of several limitations, such as relatively small memory size.

In this paper, we demonstrate how to overcome these limitations to parallel computing on GPUs with an exemplary data-intensive application, training Latent Dirichlet Allocation (LDA) models. LDA models have been successfully applied to text analysis. For large corpora, however, it takes days, even months, to train them. Our parallel approaches take the advantage of parallel computing power of GPUs and explore the algorithmic structures of LDA learning methods, therefore significantly reducing the computational cost. Furthermore, our parallel inference approaches, based on a new data partition scheme and data streaming, can be applied to not only GPUs but also any shared memory machine. Specifically, the main contributions of this paper include:

- We introduce parallel collapsed Gibbs sampling (CGS) and parallel collapsed variational Bayesian (CVB) for LDA models on GPUs. We also analyze the convergence property of the parallel variational inference and show that, with mild convexity assumptions, the parallel inference monotonically increases the variational lower bound until convergence.

- We propose a fast data partition scheme that efficiently balances the workloads across processors, fully utilizing the massive parallel mechanisms of GPUs.

- Based on this partitioning scheme, our method is also independent of specific memory consistency models: with partitioned data and parameters in exclusive memory sections, we avoid access conflict and do not sacrifice speedup caused by extra cost from a memory consistency mechanism

- We propose a data streaming scheme, which allows our methods to handle very large corpora that cannot be stored in a single GPU.

- Extensive experiments show both parallel inference algorithms on GPUs achieve the same predictive power as their sequential inference counterparts on CPUs, but significantly faster. The speedup is near linear in terms of the number of multiprocessors in the GPU card.

## 2   Latent Dirichlet Allocation

We briefly review the LDA model and two inference algorithms for LDA. [1]LDA models each of $D$ documents as a mixture over $K$ latent topics, and each topic $k$ is a multinomial distribution over a word vocabulary having $W$ distinct words denoted by $\phi_k = \{\phi_{kw}\}$, where $\phi_k$ is drawn from a symmetric Dirichlet prior with parameter $\beta$. In order to generate a document $j$, the document's mixture over topics, $\theta_j = \{\theta_{jk}\}$, is drawn from a symmetric Dirichlet prior with parameter $\alpha$ first. For the $i$th token in the document, a topic assignment $z_{ij}$ is drawn with topic $k$ chosen with probability $\theta_{jk}$. Then word $x_{ij}$ is drawn from the $z_{ij}$th topic, with $x_{ij}$ taking on value $w$ with probability $\phi_{z_{ij}w}$. Given the training data with $N$ words $\mathbf{x} = \{x_{ij}\}$, we need to compute the posterior distribution over the latent variables.

Collapsed Gibbs sampling [4] is an efficient procedure to sample the posterior distribution of topic assignment $\mathbf{z} = \{z_{ij}\}$ by integrating out all $\theta_{jk}$ and $\phi_{kw}$. Given the current state of all but one variable $z_{ij}$, the conditional distribution of $z_{ij}$ is

$$P(z_{ij} = k | \mathbf{z}^{\neg ij}, \mathbf{x}, \alpha, \beta) \propto \frac{n_{x_{ij}k}^{\neg ij} + \beta}{n_k^{\neg ij} + W\beta}(n_{jk}^{\neg ij} + \alpha) \tag{1}$$

where $n_{wk}$ denotes the number of tokens with word $w$ assigned to topic $k$, $n_{jk}$ denotes the number of tokens in document $j$ assigned to topic $k$ and $n_k^{\neg ij} = \sum_w n_{wk}^{\neg ij}$. Superscript $\neg ij$ denotes that the variable is calculated as if token $x_{ij}$ is removed from the training data.

CGS is very efficient because the variance is greatly reduced by sampling in a collapsed state space. Teh et al. [9] applied the same state space to variational Bayesian and proposed the collapsed variational Bayesian inference algorithm. It has been shown that CVB has a theoretically tighter variational bound than standard VB. In CVB, the posterior of $\mathbf{z}$ is approximated by a factorized posterior $q(\mathbf{z}) = \prod_{ij} q(z_{ij}|\gamma_{ij})$ where $q(z_{ij}|\gamma_{ij})$ is multinomial with variational parameter

$\gamma_{ij} = \{\gamma_{ijk}\}$. The inference task is to find variational parameters maximizing the variational lower bound $L(q) = \sum_{\mathbf{z}} q(\mathbf{z}) \log \frac{p(\mathbf{z}, \mathbf{x} | \alpha, \beta)}{q(\mathbf{z})}$. The authors used a computationally efficient Gaussian approximation. The updating formula for $\gamma_{ij}$ is similar to the CGS updates

$$
\begin{aligned}
\gamma_{ijk} \quad &\propto \quad (E_q[n_{x_{ijk}}^{\neg ij}] + \beta)(E_q[n_{jk}^{\neg ij}] + \alpha)(E_q[n_k^{\neg ij}] + W\beta)^{-1} \\
&\exp\left(-\frac{\mathrm{Var}_q[n_{x_{ijk}}^{\neg ij}]}{2(E_q[n_{x_{ijk}}^{\neg ij}] + \beta)^2} - \frac{\mathrm{Var}_q[n_{jk}^{\neg ij}]}{2(E_q[n_{jk}^{\neg ij}] + \alpha)^2)} + \frac{\mathrm{Var}_q[n_k^{\neg ij}]}{2(E_q[n_k^{\neg ij}] + W\beta)^2}\right)
\end{aligned} \quad (2)
$$

## 3 Parallel Algorithms for LDA Training

### 3.1 Parallel Collapsed Gibbs Sampling

A natural way to parallelize LDA training is to distribute documents across $P$ processors. Based on this idea, Newman et al. [8] introduced a parallel implementation of CGS on distributed machines, called AD-LDA. In AD-LDA, $D$ documents and document-specific counts $n_{jk}$ are distributed over $P$ processors, with $\frac{D}{P}$ documents on each processor. In each iteration, every processor $p$ independently runs local Gibbs sampling with its own copy of topic-word count $n_{kw}^p$ and topic counts $n_k^p = \sum_w n_{kw}^p$ in parallel. Then a global synchronization aggregates local counts $n_{kw}^p$ to produce global counts $n_{kw}$ and $n_k$. AD-LDA achieved substantial speedup compared with single-processor CGS training without sacrificing prediction accuracy. However, it needs to store $P$ copies of topic-word counts $n_{kw}$ for all processors, which is unrealistic for GPUs with large $P$ and large datasets due to device memory space limitation. For example, a dataset having $100,000$ vocabulary words needs at least $1.4$ GBytes to store 256-topic $n_{wk}$ for 60 processors, exceeding the device memory capacity of current high-end GPUs. In order to address this issue, we develop parallel CGS algorithm that only requires one copy of $n_{kw}$.

---

**Algorithm 1**: Parallel Collapsed Gibbs Sampling

**Input**: Word tokens $\mathbf{x}$, document partition $J_1, \ldots, J_P$ and vocabulary partition $V_1, \ldots, V_P$

**Output**: $n_{jk}, n_{wk}, z_{ij}$

1 Initialize topic assignment to each word token, set $n_k^p \leftarrow n_k$
2 **repeat**
3   **for** $l = 0$ **to** $P - 1$ **do**
    /* Sampling step      */
4     **for** *each processor $p$ in parallel* **do**
5       Sample $z_{ij}$ for $j \in J_p$ and $x_{ij} \in V_{p\oplus l}$ (Equation (1)) with global counts $n_{wk}$, global counts $n_{jk}$ and local counts $n_k^p$
6     **end**
    /* Synchronization step   */
7     Update $n_k^p$ according to Equation (3)
8   **end**
9 **until** *convergence*

---

Our parallel CGS algorithm is motivated by the following observation: for word token $w_1$ in document $j_1$ and word token $w_2$ in document $j_2$, if $w_1 \neq w_2$ and $j_1 \neq j_2$, simultaneous updates of topic assignment by (1) have no memory read/write conflicts on document-topic counts $n_{jk}$ and topic-word counts $n_{wk}$. The algorithmic flow is summarized in Algorithm 1. In addition to dividing all documents $J = \{1, \ldots, D\}$ to $P$ (disjoint) sets of documents $J_1, \ldots, J_P$ and distribute them to $P$ processors, we further divide the vocabulary words $V = \{1, \ldots, W\}$ into $P$ disjoint subsets $V_1, \ldots, V_P$, and each processor $p$ ($p = 0, \ldots, P-1$) stores a local copy of topic counts $n_k^p$. Every parallel CGS training iteration consists of $P$ *epochs*, and each epoch consists of a sampling step and a synchronization step. In the sampling step of the $l$th epoch ($l = 0, \ldots, P - 1$), processor $p$ samples topic assignments $z_{ij}$ whose document index is $j \in J_p$ and word index is $x_{ij} \in V_{p \oplus l}$. The $\oplus$ is the modulus $P$ addition operation defined by

$$a \oplus b = (a + b) \mod P,$$

and all processors run the sampling simultaneously without memory read/write conflicts on the global counts $n_{jk}$ and $n_{wk}$. Then the synchronization step uses (3) to aggregate $n_k^p$ to global counts $n_k$, which are used as local counts in the next epoch.

$$
n_k \leftarrow n_k + \sum_p (n_k^p - n_k), \qquad n_k^p \leftarrow n_k \quad (3)
$$

Our parallel CGS can be regarded as an extension to AD-LDA by using the data partition in local sampling and inserting $P-1$ more synchronization steps within an iteration. Since our data partition guarantees that any two processors access neither the same document nor the same word in an epoch, the synchronization of $n_{wk}$ in AD-LDA is equivalent to keeping $n_{wk}$ unchanged after the sampling step of the epoch. Becasue $P$ processors concurrently sample new topic assignments in parallel CGS, we don't necessarily sample from the correct posterior distribution. However, we can view it as a stochastic optimization method that maximizes $p(z|x, \alpha, \beta)$. A justification of this viewpoint can be found in [8].

### 3.2 Parallel Collapsed Variational Bayesian

The collapsed Gibbs sampling and the collapsed variational Bayesian inference [9] are similar in their algorithmic structures. As pointed out by Asuncion et al. [2], there are striking similarities between CGS and CVB. A single iteration of our parallel CVB also consists of $P$ epochs, and each epoch has an updating step and a synchronization step. The updating step updates variational parameters in a similar manner as the sampling step of parallel CGS. Counts in CGS are replaced by expectations and variances, and new variational parameters are computed by (2). The synchronization step involves an affine combination of the variational parameters in the natural parameter space.

Since multinomial distribution belongs to the exponential family, we can represent the multinomial distribution over $K$ topics defined by mean parameter $\gamma_{ij}$ in natural parameter $\lambda_{ij} = (\lambda_{ijk})$ by $\lambda_{ijk} = \log(\frac{\gamma_{ijk}}{1-\sum_{k' \neq K} \gamma_{ijk'}})$ for $k = 1, 2, \ldots, K-1$, and the domain of $\lambda_{ij}$ is unconstrained. Thus maximizing $L(q(\boldsymbol{\lambda}))$ becomes an unconstrained optimization problem. Denote $\boldsymbol{\lambda}_m = (\lambda_{ij})_{j \in J_m}$, $\boldsymbol{\lambda} = (\boldsymbol{\lambda}_0, \ldots, \boldsymbol{\lambda}_{P-1})$, $\boldsymbol{\lambda}^{new}$ and $\boldsymbol{\lambda}^{old}$ to be the variational parameters immediately after and before the updating step respectively. Let $\boldsymbol{\lambda}^{(p)} = (\boldsymbol{\lambda}_0^{old}, \ldots, \boldsymbol{\lambda}_p^{new}, \ldots, \boldsymbol{\lambda}_{P-1}^{old})$. We pick a $\boldsymbol{\lambda}^{sync}$ as the updated $\boldsymbol{\lambda}$ from a one-parameter class of variational parameters $\boldsymbol{\lambda}(\mu)$ that combines the contribution from all processors

$$\boldsymbol{\lambda}(\mu) = \boldsymbol{\lambda}^{old} + \mu \sum_{i=0}^{P-1} (\boldsymbol{\lambda}^{(i)} - \boldsymbol{\lambda}^{old}), \ \mu \geq 0.$$

Two special cases are of interest: 1) $\boldsymbol{\lambda}^{sync} = \boldsymbol{\lambda}(\frac{1}{P})$ is a convex combination of $\{\boldsymbol{\lambda}^{(p)}\}$; and 2) $\boldsymbol{\lambda}^{sync} = \boldsymbol{\lambda}(1) = \boldsymbol{\lambda}^{new}$. If (quasi)concavity [3] holds in sufficient large neighborhoods of the sequence of $\boldsymbol{\lambda}(\mu)$, say near a local maximum having negatively defined Hessian, then $L(q(\boldsymbol{\lambda}(\mu))) \geq \min_p L(q(\boldsymbol{\lambda}^{(p)})) \geq L(q(\boldsymbol{\lambda}^{old}))$ and $L(q)$ converge locally. For the second case, we keep $\gamma^{new}$ and only update $E_q[n_k]$ and $\text{Var}_q[n_k]$ similarly as (3) in the synchronization step. The formulas are

$$E[n_k] \leftarrow E[n_k] + \sum_p (E[n_k^p] - E[n_k]), \qquad E[n_k^p] \leftarrow E[n_k]$$
$$\text{Var}[n_k] \leftarrow \text{Var}[n_k] + \sum_p (\text{Var}[n_k^p] - \text{Var}[n_k]), \qquad \text{Var}[n_k^p] \leftarrow \text{Var}[n_k] \qquad (4)$$

Also, $\boldsymbol{\lambda}(1)$ assigns a larger step size to the direction $\sum_{i=0}^{P-1} (\boldsymbol{\lambda}^{(i)} - \boldsymbol{\lambda}^{old})$. Thus we can achieve a faster convergence rate if it is an ascending direction. It should be noted that our choice of $\boldsymbol{\lambda}^{sync}$ doesn't guarantee global convergence, but we shall see that $\boldsymbol{\lambda}(1)$ can produce models that have almost the same predictive power and variational lower bounds as the single-processor CVB.

### 3.3 Data Partition

In order to achieve maximal speedup, we need the partitions producing balanced workloads across processors, and we also hope that generating the data partition consumes a small fraction of time in the whole training process.

In order to present in a unified way, we define the co-occurrence matrix $R = (r_{jw})$ as: For parallel CGS, $r_{jw}$ is the number of occurrences of word $w$ in document $j$; for parallel CVB, $r_{jw} = 1$ if $w$ occurs at least once in $j$, otherwise $r_{jw} = 0$. We define the submatrix $R_{mn} = (r_{jw}) \forall j \in J_m, w \in V_n$. The optimal data partition is equivalent to minimizing the following cost function

$$C = \sum_{l=0}^{P-1} \max_{\substack{(m,n): \\ m \oplus l = n}} \{C_{mn}\}, \qquad C_{mn} = \sum_{r_{jw} \in R_{mn}} r_{jw} \qquad (5)$$

The *basic operation* in the proposed algorithms is either sampling topic assignments (in CGS) or updating variational parameters (in CVB). Each value of $l$ in the first summation term in (5) is associated with one epoch. All $R_{mn}$ satisfying $m \oplus l = n$ are the $P$ submatrices of $R$ whose entries are used to perform basic operations in epoch $l$. The number of these two types of basic operations on each unique document/word pair $(j, w)$ are all $r_{jw}$. So the total number of basic operations in $R_{m,n}$ is $C_{mn}$ for a single processor. Since all processors have to wait for the slowest processor to complete its job before a synchronization step, the maximal $C_{mn}$ is the number of basic operations for the slowest processor. Thus the total number of basic operations is $C$. We define *data partition efficiency*, $\eta$, for a given row and column partitions by

$$\eta = \frac{C_{opt}}{C}, \quad C_{opt} = \sum_{j \in J, w \in V} r_{jw}/P \tag{6}$$

where $C_{opt}$ is the theoretically minimal number of basic operations. $\eta$ is defined to be less than or equal to 1. The higher the $\eta$, the better the partitions. Exact optimization of (5) can be achieved through solving an equivalent integer programming problem. Since integer programming is NP-hard in general, and the large number of free variables for real-world datasets makes it intractable to solve, we use a simple approximate algorithm to perform data partitioning. In our observation, it works well empirically.

Here we use the convention of initial value $j_0 = w_0 = 0$. Our data partition algorithm divides row index $J$ into disjoint subsets $J_m = \{j_{(m-1)}, \ldots, j_m\}$, where $j_m = \arg\min_{j'} |mC_{opt} - \sum_{j \leq j'} r_{jw}|$. Similarly, we divide column index $V$ into disjoint subsets $V_n = \{w_{(n-1)} + 1, \ldots, w_n\}$ by $w_n = \arg\min_{w'} |mC_{opt} - \sum_{w \leq w'} r_{jw}|$. This algorithm is fast, since it needs only one full sweep over all word tokens or unique document/word pairs to calculate $j_m$ and $w_n$. In practice, we can run this algorithm for several random permutations of $J$ or $V$, and take the partitions with the highest $\eta$.

We empirically obtained high $\eta$ on large datasets with the approximate algorithm. For a word token $x$ in the corpus, the probability that $x$ is the word $w$ is $P(x = w)$, the probability that $x$ is in document $j$ is $P(x \text{ in } j)$. If we assume these two distributions are independent and $x$ is i.i.d., then for a fixed $P$, the law of large numbers asserts $P(x \text{ in } J_m) \approx \frac{j_m - j_{(m-1)}}{D} \approx \frac{1}{P}$ and $P(x \in V_n) \approx \frac{w_n - w_{(n-1)}}{W} \approx \frac{1}{P}$. Independence gives $E[C_{mn}] \approx \frac{C_{opt}}{P}$ where $C_{mn} = \sum_x \mathbf{1}_{\{x \text{ in} J_m, x \in V_n\}}$. Furthermore, the law of large numbers and the central limit theorem also give $C_{mn} \approx \frac{C_{opt}}{P}$ and the distribution of $C_{mn}$ is approximately a normal distribution. Although independence and i.i.d. assumptions are not true for real data, the above analysis holds in an approximate way. Actually, when $P = 10$, the $C_{mn}$ of NIPS and NY Times datasets (see Section 4) accepted the null hypothesis of Lilliefors' normality test with a 0.05 significance level.

## 3.4 GPU Implementation and Data Streaming

We used a Leatek Geforce 280 GTX GPU (G280) in this experiment. The G280 has 30 on-chip multiprocessors running at 1296 MHz, and each multiprocessor has 8 thread processors that are responsible for executing all threads deployed on the multiprocessor in parallel. The G280 has 1 GBytes on-board device memory, the memory bandwidth is 141.7 GB/s. We adopted NVidia's Compute Unified Device Architecture (CUDA) as our GPU programming environment. CUDA programs run in a Single Program Multiple Threads (SPMT) fashion. All threads are divided into equal-sized *thread blocks*. Threads in the same thread block are executed on a multiprocessor, and a multiprocessor can execute a number of thread blocks. We map a "processor" in the previous algorithmic description to a thread block. For a word token, fine parallel calculations, such as (1) and (2), are realized by parallel threads inside a thread block.

Given the limited amount of device memory on GPUs, we cannot load all training data and model parameters into the device memory for large-scale datasets. However, the sequential nature of Gibbs sampling and variational Bayesian inferences allow us to implement a data streaming [5] scheme which effectively reduces GPU device memory space requirements. Temporal data and variables, $x_{ij}$, $z_{ij}$ and $\gamma_{ij}$, are sent to a working space on GPU device memory on-the-fly. Computation and data transfer are carried out simultaneously, i.e. data transfer latency is hidden by computation.

| dataset | KOS | NIPS | NYT |
|---|---|---|---|
| Number of documents, D | $3,430$ | $1,500$ | $300,000$ |
| Number of words, W | $6,906$ | $12,419$ | $102,660$ |
| Number of word tokens, N | $467,714$ | $1,932,365$ | $99,542,125$ |
| Number of unique document/word pairs, M | $353,160$ | $746,316$ | $69,679,427$ |

Table 1: datasets used in the experiments.

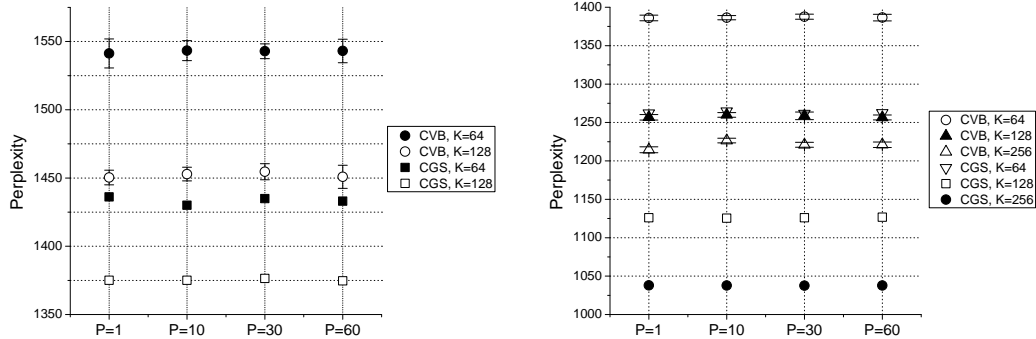

Figure 1: Test set perplexity versus number of processors $P$ for KOS (left) and NIPS (right).

## 4  Experiments

We used three text datasets retrieved from the UCI Machine Learning Repository[2] for evaluation. Statistical information about these datasets is shown in Table 4. For each dataset, we randomly extracted $90\%$ of all word tokens as the training set, and the remaining $10\%$ of word tokens are the test set. We set $\alpha = 50/K$ and $\beta = 0.1$ in all experiments [4]. We use $\boldsymbol{\lambda}^{sync} = \boldsymbol{\lambda}(1)$ in the parallel CVB, and this setting works well in all of our experiments.

### 4.1  Perplexity

We measure the performance of the parallel algorithms using test set perplexity. Test set perplexity is defined as $\exp(-\frac{1}{N^{\text{test}}}\log p(\mathbf{x}^{\text{test}}))$. For CVB, test set likelihood $p(\mathbf{x}^{\text{test}})$ is computed as

$$p(\mathbf{x}^{\text{test}}) = \prod_{ij}\log\sum_k \bar{\theta}_{jk}\bar{\phi}_{x_{ij}k} \qquad \bar{\theta}_{jk} = \frac{\alpha + E[n_{jk}]}{K\alpha + \sum_k E[n_{jk}]} \qquad \bar{\phi}_{wk} = \frac{\beta + E[n_{wk}]}{W\beta + E[n_k]} \qquad (7)$$

We report the average perplexity and the standard deviation of 10 randomly initialized runs for the parallel CVB. The typical burn-in period of CGS is about 200 iterations. We compute the likelihood $p(\mathbf{x}^{\text{test}})$ for CGS by averaging $S = 10$ samples at the end of 1000 iterations from different chains.

$$p(\mathbf{x}^{\text{test}}) = \prod_{ij}\log\frac{1}{S}\sum_s\sum_k \hat{\theta}_{jk}^s\hat{\phi}_{x_{ij}k}^s \qquad \hat{\theta}_{jk}^s = \frac{\alpha + n_{jk}^s}{K\alpha + \sum_k n_{jk}^s} \qquad \hat{\phi}_{wk}^s = \frac{\beta + n_{wk}^s}{W\beta + n_k^s} \qquad (8)$$

Two small datasets, KOS and NIPS, are used in the perplexity experiment. We computed test perplexity for different values of $K$ and $P$. Figure 1 shows the test set perplexity on KOS (left) and NIPS (right). We used the CPU to compute perplexity for $P = 1$ and the GPU for $P = 10, 30, 60$. For a fixed number of $K$, there is no significant difference between the parallel and the single-processor algorithms. It suggests our parallel algorithms converge to models having the same predictive power in terms of perplexity as single-processor LDA algorithms.

Perplexity as a function of iteration number for parallel CGS and parallel CVB on NIPS are shown in Figure 2 (a) and (b) respectively. Since CVB actually maxmizes the variational lower bound $L(q)$ on the training set, so we also investigated the convergence rate of the variational lower bound. The variational lower bound is computed using an exact method suggested in [9]. Figure 2 (c) shows the per word token variational lower bound as a function of iteration for $P = 1, 10, 30$ on a sampled

subset of KOS ($K = 64$). Both parallel algorithms converge as rapidly as the single-processor LDA algorithms. Therefore, when $P$ gets larger, convergence rate does not curtail the speedup. We surmise that these results in Figure 2 may be due to frequent synchronization and relative big step sizes in our algorithms. In fact, as we decreased the number of synchronizations in the parallel CVB, the result became significantly worse. The curve "$\mu$=1/P, P=10" in Figure 2 (right) was obtained by setting $\boldsymbol{\lambda}^{sync} = \boldsymbol{\lambda}(\frac{1}{P})$. It converged considerably slower than the other curves because of its small step size.

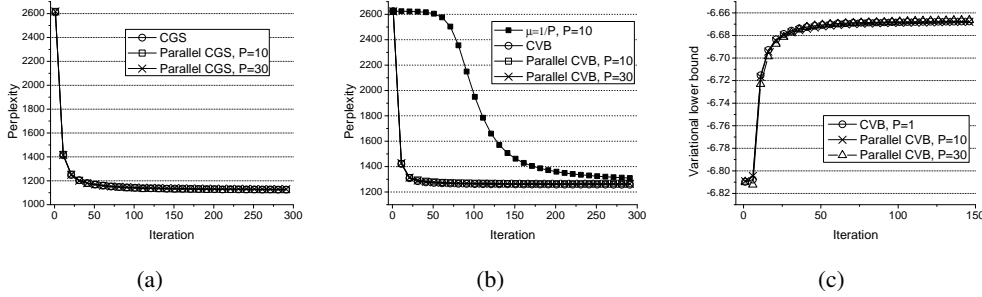

|  |  |  |
|---|---|---|
| (a) | (b) | (c) |

Figure 2: (a) Test set perplexity as a function of iteration number for the parallel CGS on NIPS, $K = 256$. (b) Test set perplexity as a function of iteration number for the parallel CVB on NIPS, $K = 128$. (c) Variational lower bound on a dataset sampled from KOS, $K = 64$.

## 4.2 Speedup

The speedup is compared with a PC equipped with an Intel quad-core 2.4GHz CPU and 4 GBytes memory. Only one core of the CPU is used. All CPU implementations are compiled by Microsoft C++ compiler 8.0 with -O2 optimization. We did our best to optimize the code through experiments, such as using better data layout and reducing redundant computation. The final CPU code is almost twice as fast as the initial code.

Our speedup experiments are conducted on the NIPS dataset for both parallel algorithms and the large NYT dataset for only the parallel CGS, because $\gamma_{ij}$ of the NYT dataset requires too much memory space to fit into our PC's host memory. We measure the speedup on a range of $P$ with or without data streaming. As the baseline, average running times on the CPU are: 4.24 seconds on NIPS ($K = 256$) and 22.1 seconds on NYT ($K = 128$) for the parallel CGS, and 11.1 seconds ($K = 128$) on NIPS for the parallel CVB. Figure 3 shows the speedup of the parallel CGS (left) and the speedup of the parallel CVB (right) with the data partition efficiency $\eta$ under the speedup. We note that when $P > 30$, more threads are deployed on a multiprocessor. Therefore data transfer between the device memory and the multiprocessor is better hidden by computation on the threads. As a result, we have extra speedup when the number of "processors" (thread blocks) is larger than the number of multiprocessors on the GPU.

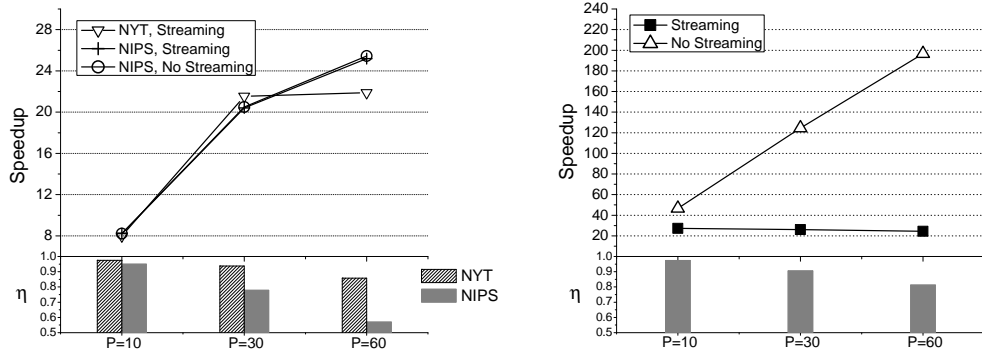

Figure 3: Speedup of parallel CGS (left) on NIPS and NYT, and speedup of parallel CVB (right) on NIPS. Average running times on the CPU are 4.24 seconds on NIPS and 22.1 seconds on NYT for the parallel CGS, and 11.1 seconds on NIPS for the parallel CVB, respectively. Although using data streaming reduces the speedup of parallel CVB due to the low bandwidth between the PC host memory and the GPU device memory, it enables us to use a GPU card to process large-volume data.

The synchronization overhead is very small since $P \ll N$ and the speedup is largely determined by the maximal number of nonzero elements in all partitioned submatrices. As a result, the speedup (when not using data streaming) is proportional to $\eta P$. The bandwidth between the PC host memory and the GPU device memory is $\sim 1.0$ GB/s, which is higher than the computation bandwidth (size of data processed by the GPU per second) of the parallel CGS. Therefore, the speedup with or without data streaming is almost the same for the parallel CGS. But the speedup with or without data streaming differs dramatically for the parallel CVB, because its computation bandwidth is roughly $\sim 7.2$ GB/s for $K = 128$ due to large memory usage of $\gamma_{ij}$, higher than the maximal bandwidth that data streaming can provide. The high speedup of the parallel CVB without data streaming is due to a hardware supported exponential function and a high performance implementation of *parallel reduction* that is used to normalize $\gamma_{ij}$ calculated from (2). Figure 3 (right) shows that the larger the $P$, the smaller the speedup for the parallel CVB with data streaming. The reason is when $P$ becomes large, the data streaming management becomes more complicated and introduces more latencies on data transfer.

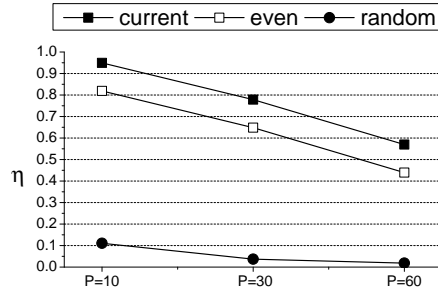

Figure 4: data partition efficiency $\eta$ of various data partition algorithms for $P = 10, 30, 60$. Due to the negligible overheads for the synchronization steps, the speedup is proportional to $\eta$ in practice.

Figure 4 shows data partition efficiency $\eta$ of various data partition algorithms for $P = 10, 30, 60$ on NIPS. "current" is the data partition algorithm proposed in section 3.3, "even" partitions documents and word vocabulary into roughly equal-sized subsets by setting $j_m = \lfloor \frac{mD}{P} \rfloor$ and $w_n = \lfloor \frac{nW}{P} \rfloor$. "random" is a data partition obtained by randomly partitioning documents and words. We see that the proposed data partition algorithm outperforms the other algorithms.

More than 20x speedup is achieved for both parallel algorithms with data streaming. The speedup of the parallel CGS enables us to run 1000 iterations (K=128) Gibbs sampling on the large NYT dataset within 1.5 hours, and it yields the same perplexity 3639 ($S = 5$) as the result obtained from 30-hour training on a CPU.

## 5 Related Works and Discussion

Our work is closely related to several previous works, including the distributed LDA by Newman et al. [8], asynchronous distributed LDA by Asuncion et al. [1] and the parallelized variational EM algorithm for LDA by Nallapati et al. [7]. For these works LDA training was parallelized on distributed CPU clusters and achieved impressive speedup. Unlike their works, ours shows how to use GPUs to achieve significant, scalable speedup for LDA training while maintaining correct, accurate predictions.

Masada et al. recently proposed a GPU implementation of CVB [6]. Masada et al. keep one copy of $n_{wk}$ while simply maintaining the same algorithmic structure for their GPU implementation as Newman et al. did on a CPU cluster. However, with the limited memory size of a GPU, compared to that of a CPU cluster, this can lead to memory access conflicts. This issue becomes severe when one performs many parallel jobs (threadblocks) and leads to wrong inference results and operation failure, as reported by Masada et al. Therefore, their method is not easily scalable due to memory access conflicts. Different from their approach, ours are scalable with more multiprocessors with the the proposed partitioning scheme and data streaming. They can also be used as general techniques to parallelize other machine learning models that involve sequential operations on matrix, such as online training of matrix factorization.

**Acknowledgements**

We thank Max Welling and David Newman for providing us with the link to the experimental data. We also thank the anonymous reviewers, Dong Zhang and Xianxing Zhang for their invaluable inputs. F. Yan conducted this research at Microsoft Research Asia. F. Yan and Y. Qi were supported by NSF IIS-0916443 and Microsoft Research.

## Footnotes

[1]We use indices to represent topics, documents and vocabulary words.

[2]http://archive.ics.uci.edu/ml/datasets/Bag+of+Words

# References

[1] A. Asuncion, P. Smyth, and M. Welling. Asynchronous distributed learning of topic models. In D. Koller, D. Schuurmans, Y. Bengio, and L. Bottou, editors, *NIPS*, pages 81–88. MIT Press, 2008.

[2] A. Asuncion, M. Welling, P. Smyth, and Y. W. Teh. On smoothing and inference for topic models. In *Proceedings of the International Conference on Uncertainty in Artificial Intelligence*, 2009.

[3] S. Boyd and L. Vandenberghe. *Convex Optimization*. Cambridge University Press, March 2004.

[4] T. L. Griffiths and M. Steyvers. Finding scientific topics. *Proceedings of the National Academy Science*, 101 (suppl. 1):5228–5235, April 2004.

[5] F. Labonte, P. Mattson, W. Thies, I. Buck, C. Kozyrakis, and M. Horowitz. The stream virtual machine. In *PACT '04: Proceedings of the 13th International Conference on Parallel Architectures and Compilation Techniques*, pages 267–277, Washington, DC, USA, 2004. IEEE Computer Society.

[6] T. Masada, T. Hamada, Y. Shibata, and K. Oguri. Accelerating collapsed variational bayesian inference for latent Dirichlet allocation with Nvidia CUDA compatible devices. In *IEA-AIE*, 2009.

[7] R. Nallapati, W. Cohen, and J. Lafferty. Parallelized variational EM for latent Dirichlet allocation: An experimental evaluation of speed and scalability. 2007.

[8] D. Newman, A. Asuncion, P. Smyth, and M. Welling. Distributed inference for latent Dirichlet allocation. In *NIPS*, 2007.

[9] Y. W. Teh, D. Newman, and M. Welling. A collapsed variational Bayesian inference algorithm for Latent Dirichlet allocation. In B. Schölkopf, J. C. Platt, and T. Hoffman, editors, *NIPS*, pages 1353–1360. MIT Press, 2006.

